# An Incremental Nearest Neighbor Algorithm with Queries

**Joel Ratsaby***
N.A.P. Inc.
Hollis, New York

## Abstract

We consider the general problem of learning multi-category classification from labeled examples. We present experimental results for a nearest neighbor algorithm which actively selects samples from different pattern classes according to a querying rule instead of the *a priori* class probabilities. The amount of improvement of this query-based approach over the passive batch approach depends on the complexity of the Bayes rule. The principle on which this algorithm is based is general enough to be used in any learning algorithm which permits a model-selection criterion and for which the error rate of the classifier is calculable in terms of the complexity of the model.

## 1  INTRODUCTION

We consider the general problem of learning multi-category classification from labeled examples. In many practical learning settings the time or sample size available for training are limited. This may have adverse effects on the accuracy of the resulting classifier. For instance, in learning to recognize handwritten characters typical time limitation confines the training sample size to be of the order of a few hundred examples. It is important to make learning more efficient by obtaining only training data which contains significant information about the separability of the pattern classes thereby letting the learning algorithm participate actively in the sampling process. Querying for the class labels of specificly selected examples in the input space may lead to significant improvements in the generalization error (cf. Cohn, Atlas & Ladner, 1994, Cohn, 1996). However in learning pattern recognition this is not always useful or possible. In the handwritten recognition problem, the computer could ask the user for labels of selected patterns generated by the computer

however labeling such patterns are not necessarily representative of his handwriting style but rather of his reading recognition ability. On the other hand it is possible to let the computer (learner) select particular pattern classes, not necessarily according to their *a priori* probabilities, and then obtain randomly drawn patterns according to the underlying unknown class-conditional probability distribution. We refer to such selective sampling as *sample querying*. Recent theory (cf. Ratsaby, 1997) indicates that such freedom to select different classes at any time during the training stage is beneficial to the accuracy of the classifier learnt. In the current paper we report on experimental results for an incremental algorithm which utilizes this sample-querying procedure.

## 2 THEORETICAL BACKGROUND

We use the following setting: Given $M$ distinct pattern classes each with a class conditional probability density $f_i(x)$, $1 \leq i \leq M$, $x \in \mathbb{R}^d$, and *a priori* probabilities $p_i$, $1 \leq i \leq M$. The functions $f_i(x)$, $1 \leq i \leq M$, are assumed to be unknown while the $p_i$ are assumed to be known or easily estimable as is the case of learning character recognition. For a sample-size vector $m = [m_1, \ldots, m_M]$ where $\sum_{i=1}^{M} m_i = \overline{m}$ denote by $\zeta^m = \{(x_j, y_j)\}_{j=1}^{\overline{m}}$ a sample of labeled examples consisting of $m_i$ example from pattern class $i$ where $y_j$, $1 \leq j \leq \overline{m}$, are chosen *not* necessarily at random from $\{1, 2, \ldots, M\}$, and the corresponding $x_j$ are drawn at random i.i.d. according to the class conditional probability density $f_{y_j}(x)$. The expected misclassification error of a classifier $c$ is referred to as the *loss* of $c$ and is denoted by $L(c)$. It is defined as the probability of misclassification of a randomly drawn $x$ with respect to the underlying mixture probability density function $f(x) = \sum_{i=1}^{M} p_i f_i(x)$. The loss is commonly represented as $L(c) = \mathrm{E}1_{\{x:c(x) \neq y(x)\}}$, where $1_{\{x \in A\}}$ is the indicator function of a set $A$, expectation is taken with respect to the joint probability distribution $f_y(x)p(y)$ where $p(y)$ is a discrete probability distribution taking values $p_i$ over $1 \leq i \leq M$, while $y$ denotes the label of the class whose distribution $f_y(x)$ was used to draw $x$. The loss $L(c)$ may also be written as $L(c) = \sum_{i=1}^{M} p_i \mathrm{E}_i 1_{\{c(x) \neq i\}}$ where $\mathrm{E}_i$ denotes expectation with respect to $f_i(x)$. The pattern recognition problem is to learn based on $\zeta^m$ the optimal classifier, also known as the *Bayes classifier*, which by definition has minimum loss which we denote by $L^*$.

A multi-category classifier $c$ is represented as a vector $c(x) = [c_1(x), \ldots, c_M(x)]$ of *boolean classifiers*, where $c_i(x) = 1$ if $c(x) = i$, and $c_i(x) = 0$ otherwise, $1 \leq i \leq M$. The loss $L(c)$ of a multi-category classifier $c$ may then be expressed as the average of the losses of its component classifiers, i.e., $L(c) = \sum_{i=1}^{M} p_i L(c_i)$ where for a boolean classifier $c_i$ the loss is defined as $L(c_i) = \mathrm{E}_i 1_{\{c_i(x) \neq 1\}}$. As an estimate of $L(c)$ we define the *empirical loss* $L_m(c) = \sum_{i=1}^{M} p_i L_{m_i}(c)$ where $L_{m_i}(c) = \frac{1}{m_i} \sum_{j:y_j=i} 1_{\{c(x_j) \neq i\}}$ which may also can be expressed as $L_{m_i}(c_i) = \frac{1}{m_i} \sum_{j:y_j=i} 1_{\{c_i(x_j) \neq 1\}}$.

The family of all classifiers is assumed to be decomposed into a multi-structure $S = S_1 \times S_2 \times \cdots \times S_M$, where $S_i$ is a nested structure (cf. Vapnik, 1982) of boolean families $\mathcal{B}_{k_{j_i}}$, $j_i = 1, 2, \ldots$, for $1 \leq i \leq M$, i.e., $S_1 = \mathcal{B}_{k_1}, \mathcal{B}_{k_2}, \ldots, \mathcal{B}_{k_{j_1}}, \ldots$, $S_2 = \mathcal{B}_{k_1}, \mathcal{B}_{k_2}, \ldots, \mathcal{B}_{k_{j_2}}, \ldots$, up to $S_M = \mathcal{B}_{k_1}, \mathcal{B}_{k_2}, \ldots, \mathcal{B}_{k_{j_M}}, \ldots$, where $k_{j_i} \in \mathbb{Z}_+$ denotes the VC-dimension of $\mathcal{B}_{k_{j_i}}$ and $\mathcal{B}_{k_{j_i}} \subseteq \mathcal{B}_{k_{j_i+1}}$, $1 \leq i \leq M$. For any fixed positive integer vector $j \in \mathbb{Z}_+^M$ consider the class of vector classifiers $\mathcal{H}_{k(j)} = \mathcal{B}_{k_{j_1}} \times \mathcal{B}_{k_{j_2}} \times \cdots \times \mathcal{B}_{k_{j_M}} \equiv \mathcal{H}_k$ where we take the liberty in dropping the multi-index $j$ and write $k$ instead of $k(j)$. Define by $\mathcal{G}_k$ the subfamily of $\mathcal{H}_k$ consisting

of classifiers $c$ that are well-defined, i.e., ones whose components $c_i$, $1 \leq i \leq M$ satisfy $\bigcup_{i=1}^{M}\{x : c_i(x) = 1\} = \mathbb{R}^d$ and $\{x : c_i(x) = 1\}\bigcap\{x : c_j(x) = 1\} = \emptyset$, for $1 \leq i \neq j \leq M$.

From the Vapnik-Chervonenkis theory (cf. Vapnik, 1982, Devroye, Gyorfi & Lugosi, 1996) it follows that the loss of any boolean classifier $c_i \in \mathcal{B}_{k_{j_i}}$ is, with high confidence, related to its empirical loss as $L(c_i) \leq L_{m_i}(c_i) + \epsilon(m_i, k_{j_i})$ where $\epsilon(m_i, k_{j_i}) = const \; \sqrt{k_{j_i} \ln m_i/m_i}$, $1 \leq i \leq M$, where henceforth we denote by *const* any constant which does not depend on the relevant variables in the expression. Let the vectors $m = [m_1, \ldots, m_M]$ and $k \equiv k(j) = [k_{j_1}, \ldots, k_{j_M}]$ in $\mathbb{Z}_+^M$. Define $\epsilon(m, k) = \sum_{i=1}^{M} p_i \epsilon(m_i, k_{j_i})$. It follows that the deviation between the empirical loss and the loss is bounded uniformly over all multi-category classifiers in a class $\mathcal{G}_k$ by $\epsilon(m, k)$. We henceforth denote by $c_k^*$ the optimal classifier in $\mathcal{G}_k$, i.e., $c_k^* = \text{argmin}_{c \in \mathcal{G}_k} L(c)$ and $\hat{c}_k = \text{argmin}_{c \in \mathcal{G}_k} L_m(c)$ is the empirical loss minimizer over the class $\mathcal{G}_k$.

The above implies that the classifier $\hat{c}_k$ has a loss which is no more than $L(c_k^*) + \epsilon(m, k)$. Denote by $k^*$ the minimal complexity of a class $\mathcal{G}_k$ which contains the Bayes classifier. We refer to it as the *Bayes complexity* and henceforth assume $k_i^* < \infty$, $1 \leq i \leq M$. If $k^*$ was known then based on a sample of size $\overline{m}$ with a sample size vector $m = [m_1, \ldots, m_M]$ a classifier $\hat{c}_{k^*}$ whose loss is bounded from above by $L^* + \epsilon(m, k^*)$ may be determined where $L^* = L(c_{k^*}^*)$ is the Bayes loss. This bound is minimal with respect to $k$ by definition of $k^*$ and we refer to it as the *minimal criterion*. It can be further minimized by selecting a sample of size vector $m^* = \text{argmin}_{\{m \in \mathbb{Z}_+^M : \sum_{i=1}^{M} m_i = \overline{m}\}} \epsilon(m, k^*)$. This basically says that more examples should be queried from pattern classes which require more complex discriminating rules within the Bayes classifier. Thus sample-querying via minimization of the minimal criterion makes learning more efficient through tuning the subsample sizes to the complexity of the Bayes classifier. However the Bayes classifier depends on the underlying probability distributions which in most interesting scenarios are unknown thus $k^*$ should be assumed unknown. In (Ratsaby, 1997) an incremental learning algorithm, based on Vapnik's structural risk minimization, generates a random complexity sequence $\hat{k}(n)$, corresponding to a sequence of empirical loss minimizers $\hat{c}_{\hat{k}(n)}$ over $\mathcal{G}_{\hat{k}(n)}$, which converges to $k^*$ with increasing time $n$ for learning problems with a zero Bayes loss. Based on this, a sample-query rule which achieves the same minimization is defined without the need to know $k^*$. We briefly describe the main ideas next.

At any time $n$, the criterion function is $\epsilon(\cdot, \hat{k}(n))$ and is defined over the $m$-domain $\mathbb{Z}_+^M$. A gradient descent step of a fixed size is taken to minimize the current criterion. After a step is taken, a new sample-size vector $m(n + 1)$ is obtained and the difference $m(n + 1) - m(n)$ dictates the sample-query at time $n$, namely, the increment in subsample size for each of the $M$ pattern classes. With increasing $n$ the vector sequence $m(n)$ gets closer to an *optimal path* defined as the set which is comprised of the solutions to the minimization of $\epsilon(m, k^*)$ under all different constraints of $\sum_{i=1}^{M} m_i = \overline{m}$, where $\overline{m}$ runs over the positive integers. Thus for all large $n$ the sample-size vector $m(n)$ is optimal in that it minimizes the minimal criterion $\epsilon(\cdot, k^*)$ for the current total sample size $\overline{m}(n)$. This constitutes the sample-querying procedure of the learning algorithm. The remaining part does empirical loss minimization over the current class $\mathcal{G}_{\hat{k}(n)}$ and outputs $\hat{c}_{\hat{k}(n)}$. By assumption, since the Bayes classifier is contained in $\mathcal{G}_{k^*}$, it follows that for all large $n$, the loss $L(\hat{c}_{\hat{k}(n)}) \leq L^* + \min_{\{m \in \mathbb{Z}_+^M : \sum_{i=1}^{M} m_i = \overline{m}(n)\}} \epsilon(m, k^*)$, which is basically the minimal criterion mentioned above. Thus the algorithm produces a classifier $\hat{c}_{\hat{k}(n)}$ with a

minimal loss even when the Bayes complexity $k^*$ is unknown.

In the next section we consider specific model classes consisting of nearest-neighbor classifiers on which we implement this incremental learning approach.

# 3  INCREMENTAL NEAREST-NEIGHBOR ALGORITHM

Fix and Hodges , cf. Silverman & Jones (1989). introduced the simple but powerful nearest-neighbor classifier which based on a labeled training sample $\{(x_i, y_i)\}_{i=1}^{\overline{m}}$, $x_i \in \mathbb{R}^d$, $y_i \in \{1, 2, \ldots, M\}$, when given a pattern $x$, it outputs the label $y_j$ corresponding to the example whose $x_j$ is closest to $x$. Every example in the training sample is used for this decision (we denote such an example as a *prototype*) thus the empirical loss is zero. The condensed nearest-neighbor algorithm (Hart, 1968) and the reduced nearest neighbor algorithm (Gates, 1972) are procedures which aim at reducing the number of prototypes while maintaining a zero empirical loss. Thus given a training sample of size $\overline{m}$, after running either of these procedures, a nearest neighbor classifier having a zero empirical loss is generated based on $\overline{s} \leq \overline{m}$ prototypes. Learning in this manner may be viewed as a form of empirical loss minimization with a complexity regularization component which puts a penalty proportional to the number of prototypes.

A cell boundary $e_{i,j}$ of the voronoi diagram (cf. Preparata & Shamos, 1985) corresponding to a multi-category nearest-neighbor classifier $c$ is defined as the $(d-1)$-dimensional perpendicular-bisector hyperplane of the line connecting the $x$-component of two prototypes $x_i$ and $x_j$. For a fixed $l \in \{1, \ldots, M\}$, the collection of voronoi cell-boundaries based on pairs of prototypes of the form $(x_i, l)$, $(x_j, q)$ where $q \neq l$, forms the boundary which separates the decision region labeled $l$ from its complement and represents the boolean nearest-neighbor classifier $c_l$. Denote by $k_l$ the number of such cell-boundaries and denote by $s_l$ the number of prototypes from a total of $m_l$ examples from pattern class $l$. The value of $k_l$ may be calculated directly from the knowledge of the $s_l$ prototypes, $1 \leq l \leq M$, using various algorithms. The boolean classifier $c_l$ is an element of an infinite class of boolean classifiers based on partitions of $\mathbb{R}^d$ by arrangements of $k_l$ hyperplanes of dimensionality $d-1$ where each of the cells of a partition is labeled either 0 or 1. It follows, cf. Devroye et. al. (1996), that the loss of a multi-category nearest-neighbor classifier $c$ which consists of $s_l$ prototypes out of $m_l$ examples, $1 \leq l \leq M$, is bounded as $L(c) \leq L_m(c) + \epsilon(m, k)$, where the *a priori* probabilities are taken as known, $m = [m_1, \ldots, m_M]$, $k = [k_1, \ldots, k_M]$ and $\epsilon(m, k) = \sum_{l=1}^{M} p_l \epsilon(m_l, k_l)$, where $\epsilon(m_l, k_l) = const \sqrt{((d+1)k_l \ln m_l + (ek_l/d)^d)/m_l}$. Letting $k^*$ denote the Bayes complexity then $\epsilon(\cdot, k^*)$ represents the minimal criterion.

The next algorithm uses the Condense and Reduce procedures in order to generate a sequence of classifiers $\hat{c}_{\hat{k}(n)}$ with a complexity vector $\hat{k}(n)$ which tends to $k^*$ as $n \to \infty$. A sample-querying procedure referred to as Greedy Query (GQ) chooses at any time $n$ to increment the single subsample of pattern class $j^*(n)$ where $m_{j^*(n)}$ is the direction of maximum descent of the criterion $\epsilon(\cdot, \hat{k}(n))$ at the current sample-size vector $m(n)$. For the part of the algorithm which utilizes a Delaunay-Triangulation procedure we use the fast Fortune's algorithm (cf. O'Rourke ) which can be used only for dimensionality $d = 2$. Since all we are interested is in counting Voronoi borders between all adjacent Voronoi cells then an efficient computation is possible also for dimensions $d > 2$ by resorting to linear programming for computing the adjacencies of facets of a polyhedron, cf. Fukuda (1997).

**Incremental Nearest Neighbor (INN) Algorithm**

**Initialization: (Time $n = 0$)**

Let increment-size $\Delta$ be a fixed small positive integer. Start with $m(0) = [c, \ldots, c]$, where $c$ is a small positive integer. Draw $\zeta^{m(0)} = \{\zeta^{m_j(0)}\}_{j=1}^M$ where $\zeta^{m_j(0)}$ consists of $m_j(0)$ randomly drawn i.i.d. examples from pattern class $j$.

**While (number of available examples $\geq \Delta$) Do:**

1. **Call Procedure CR**: $\hat{c}_{\hat{k}(n)} = CR(\zeta^{m(n)})$.

2. **Call Procedure GQ**: $m(n+1) = GQ(n)$.

3. $n := n + 1$.

**End While**

`//Used up all examples.`

**Output:** NN-classifier $\hat{c}_{\hat{k}(n)}$.

**Procedure Condense-Reduce (CR)**

**Input:** Sample $\zeta^{m(n)}$ stored in an array $A[]$ of size $\overline{m}(n)$.

**Initialize:** Make only the first example $A[1]$ be a prototype.

`//Condense`

**Do:**

$ChangeOccured := $ FALSE.

**For** $i = 1, \ldots, \overline{m}(n)$:

- **Classify** $A[i]$ based on available prototypes using the NN-Rule.
- **If** not correct **then**
  - Let $A[i]$ be a prototype.
  - $ChangeOccured := $ TRUE.
- **End If**

**End For**

**While** ($ChangeOccured$).

`//Reduce`

**Do:**

$ChangeOccured := $ FALSE.

**For** $i = 1, \ldots, \overline{m}(n)$:

- **If** $A[i]$ is a prototype then classify it using the remaining prototypes by the NN-Rule.
- **If** correct **then**
  - Make $A[i]$ be not a prototype.
  - $ChangeOccured := $ TRUE.
- **End If**

**End For**

**While** ($ChangeOccured$).

**Run Delaunay-Triangulation** Let $\hat{k}(n) = [\hat{k}_1, \ldots, \hat{k}_M]$, $\hat{k}_i$ denotes the number of Voronoi-cell boundaries associated with the $\hat{s}_i$ prototypes.

**Return** (NN-classifier with complexity vector $\hat{k}(n)$).

**Procedure Greedy-Query (GQ)**

**Input:** Time $n$.

$$j^*(n) := \text{argmax}_{1 \leq j \leq M} \left| \frac{\partial}{\partial m_j} \epsilon(m, \hat{k}(n)) \right|_{|_{m(n)}}$$

**Draw:** $\Delta$ new i.i.d. examples from class $j^*(n)$. Denote them by $\zeta$.

**Update Sample:** $\zeta^{m_{j^*(n)}(n+1)} := \zeta^{m_{j^*(n)}(n)} \bigcup \zeta$, while $\zeta^{m_i(n+1)} := \zeta^{m_i(n)}$, for $1 \leq i \neq j^*(n) \leq M$.

**Return:** $(m(n) + \Delta e_{j^*(n)})$, where $e_j$ is an all zero vector except 1 at $j^{th}$ element.

## 3.1 EXPERIMENTAL RESULTS

We ran algorithm INN on several two-dimensional ($d = 2$) multi-category classification problems and compared its generalization error versus total sample size $\overline{m}$ with that of batch learning, the latter uses Procedure CR (but not Procedure GQ) with uniform subsample proportions, i.e., $m_i = \frac{\overline{m}}{M}$, $1 \leq i \leq M$.

We ran three classification problems consisting of 4 equiprobable pattern classes with a zero Bayes loss. The generalization curves represent the average of 15 independent learning runs of the empirical error on a fixed size test set. Each run (both for INN and Batch learning) consists of 80 independent experiments where each differs by 10 in the sample size used for training where the maximum sample size is 800. We call an experiment a success if INN results in a lower generalization error than Batch. Let $p$ be the probability of INN beating Batch. We wish to reject the hypothesis $H$ that $p = \frac{1}{2}$ which says that INN and Batch are approximately equal in performance. The results are displayed in Figure 1 as a series of pairs, the first picture showing the pattern classes of the specific problem while the second shows the learning curves for the two learning algorithms. Algorithm INN outperformed the simple Batch approach with a reject level of less than 1%, the latter ignoring the inherent Bayes complexity and using an equal subsample size for each of the pattern classes. In contrast, the INN algorithm learns, incrementally over time, which of the classes are harder to separate and queries more from these pattern classes.

## Footnotes

*The author's coordinates are: Address: Hamered St. #2, Ra'anana, ISRAEL. Email: jer@ee.technion.ac.il

## References

Cohn D., Atlas L., Ladner R. (1994), Improving Generalization with Active Learning. *Machine Learning*, Vol 15, p.201-221.

Devroye L., Gyorfi L. Lugosi G. (1996). "A Probabilistic Theory of Pattern Recognition", Springer Verlag.

Fukuda K. (1997). Frequently Asked Questions in Geometric Computation. Technical report, Swiss Federal Institute of technology, Lausanne. Available at `ftp://ftp.ifor.ethz.ch/pub/fukuda/reports`.

Gates, G. W. (1972) The Reduced Nearest Neighbor Rule. *IEEE Trans. Info. Theo.*, p.431-433.

Hart P. E. (1968) The Condensed Nearest Neighbor Rule. *IEEE Trans. on Info. Theo.*, Vol. IT-14, No. 3.

O'rourke J. (1994). "Computational Geometry in C". Cambridge University Press.

Ratsaby, J. (1997) Learning Classification with Sample Queries. Electrical Engineering Dept., Technion, CC PUB #196. Available at URL http://www.ee.technion.ac.il/ jer/iandc.ps.

Rivest R. L., Eisenberg B. (1990), On the sample complexity of pac-learning using random and chosen examples. *Proceedings of the 1990 Workshop on Computational Learning Theory*, p. 154-162, Morgan Kaufmann, San Maeto, CA.

B. W. Silverman and M. C. Jones. E. Fix and J. l. Hodges (1951): An important contribution to nonparametric discriminant analysis and density estimation—commentary on Fix and Hodges (1951). *International statistical review*, 57(3), p.233-247, 1989.

Vapnik V.N., (1982), "Estimation of Dependences Based on Empirical Data", Springer-Verlag, Berlin.

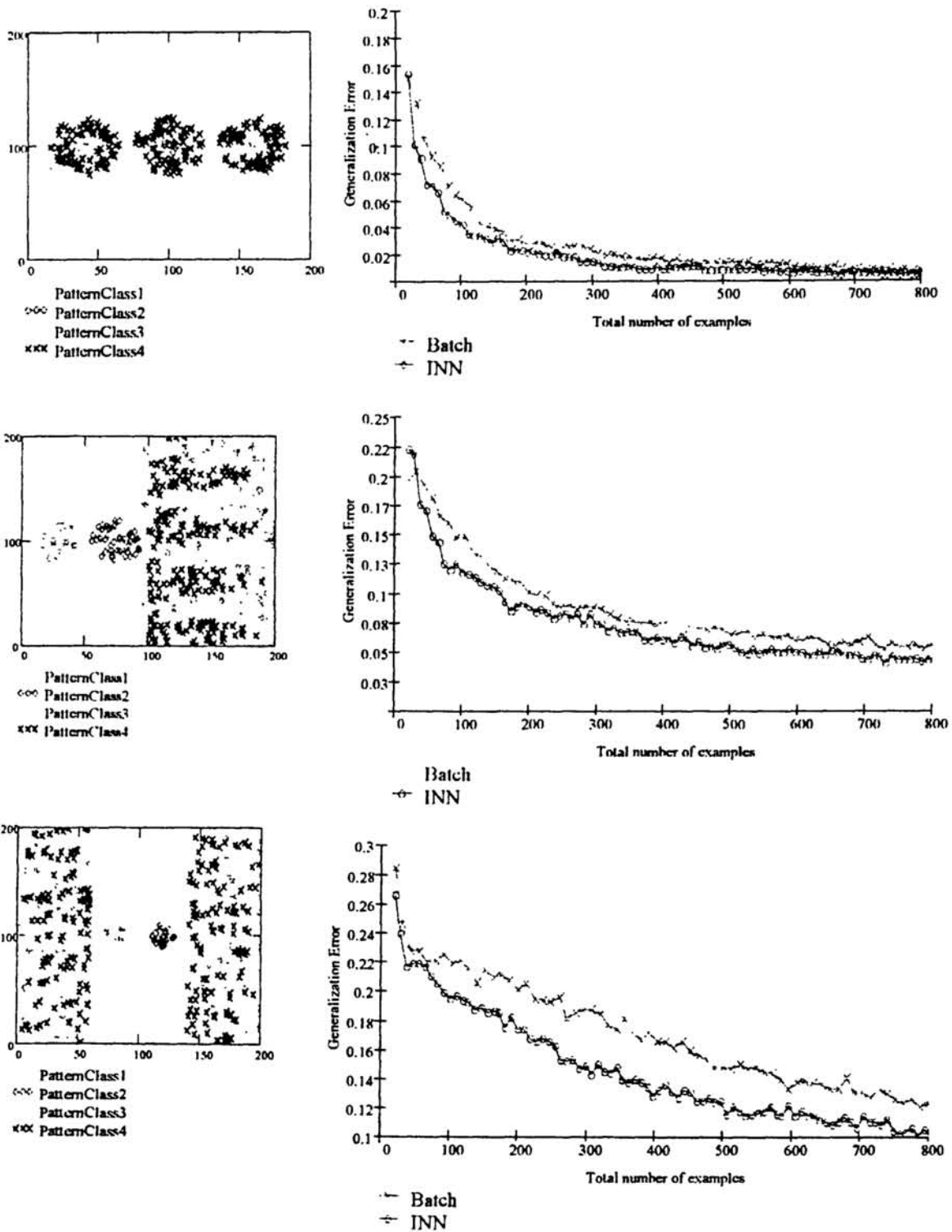

Figure 1. Three different Pattern Classification Problems and Learning
Curves of the INN-Algorithm compared to Batch Learning.